# Computing with Finite and Infinite Networks

**Ole Winther***
Theoretical Physics, Lund University
Sölvegatan 14 A, S-223 62 Lund, Sweden
`winther@nimis.thep.lu.se`

## Abstract

Using statistical mechanics results, I calculate learning curves (average generalization error) for Gaussian processes (GPs) and Bayesian neural networks (NNs) used for regression. Applying the results to learning a teacher defined by a two-layer network, I can directly compare GP and Bayesian NN learning. I find that a GP in general requires $\mathcal{O}(d^s)$-training examples to learn input features of order $s$ ($d$ is the input dimension), whereas a NN can learn the task with order the number of adjustable weights training examples. Since a GP can be considered as an infinite NN, the results show that even in the Bayesian approach, it is important to limit the complexity of the learning machine. The theoretical findings are confirmed in simulations with analytical GP learning and a NN mean field algorithm.

## 1 Introduction

Non-parametric kernel methods such as Gaussian Processes (GPs) and Support Vector Machines (SVMs) are closely related to neural networks (NNs). These may be considered as single layer networks in a possible infinite dimensional feature space. Both the Bayesian GP approach and SVMs regularize the learning problem so that only a finite number of the features (dependent on the amount of data) is used.

Neal [1] has shown that Bayesian NNs converge to GPs in the limit of infinite number of hidden units and furthermore argued that (1) there is no reason to believe that real-world problem should require only a 'small' number of hidden units and (2) there are in the Bayesian approach no reasons (besides computational) to limit the size of the network. Williams [2] has derived kernels allowing for efficient computation with both infinite feed-forward and radial basis networks.

In this paper, I show that learning with a finite rather than infinite networks can make a profound difference by studying the case where the task to be learned is defined by a large but finite two-layer NN. A theoretical analysis of the Bayesian approach to learning this task shows that the Bayesian student makes a learning transition from a linear model to specialized non-linear one when the number of examples is of the order of the number of adjustable weights in the network. This effect–which is also seen in the simulations–is a consequence of the finite complexity of the network. In an infinite network, i.e. a GP on the

other hand such a transition will not occur. It will eventually learn the task but it requires $\mathcal{O}(d^s)$-training examples to learn features of order $s$, where $d$ is the input dimension.

Here, I focus entirely on regression. However, the basic conclusions regarding learning with kernel methods and NNs turn out to be valid more generally, e.g. for classification unpublished results and [3].

I consider the usual Bayesian setup of supervised learning: A training set $D_N = \{(\mathbf{x}_i, y_i)|i = 1 \ldots, N\}$ ($\mathbf{x} \in R^d$ and $y \in R$) is known and the output for the new input $\mathbf{x}$ is predicted by the function $f(\mathbf{x})$ which is sampled from the prior distribution of model outputs. I will consider both a Gaussian process prior and the prior implied by a large (but finite) two-layer network. The output noise is taken to be Gaussian, so the Likelihood becomes $p(y|f(\mathbf{x})) = e^{-(y-f(\mathbf{x}))^2/2}/\sqrt{2\pi\sigma^2}$. The error measure is minus the log-Likelihood and Bayes regressor (which minimizes the expected error) is the posterior mean prediction

$$\langle f(\mathbf{x}) \rangle = \frac{\mathbf{E_f} f(\mathbf{x}) \prod_i p(y_i|f(\mathbf{x}_i))}{\mathbf{E_f} \prod_i p(y_i|f(\mathbf{x}_i))} , \tag{1}$$

where I have introduced $\mathbf{E_f}$, $\mathbf{f} = f(\mathbf{x}_1), \ldots, f(\mathbf{x}_N), f(\mathbf{x})$, to denote an average with respect to the model output prior.

**Gaussian processes.** In this case, the model output prior is by definition Gaussian

$$p(\mathbf{f}) = \frac{1}{\sqrt{(2\pi)^N \det \mathbf{C}}} \exp\left(-\frac{1}{2}\mathbf{f}^T \mathbf{C}^{-1}\mathbf{f}\right) , \tag{2}$$

where $\mathbf{C}$ is the covariance matrix. The covariance matrix is computed from the kernel (covariance function) $C(\mathbf{x}, \mathbf{x}')$. Below I give an explicit example corresponding to an infinite two-layer network.

**Bayesian neural networks** The output of the two-layer NN is given by $f(\mathbf{x}, \mathbf{w}, \mathbf{W}) = \frac{1}{\sqrt{K}}\sum_k^K W_k \Phi(\mathbf{w}_k \cdot \mathbf{x})$, where an especially convenient choice of transfer function in what follows is $\Phi(z) = \int_{-z}^\infty dt e^{-t^2/2}/\sqrt{2\pi}$. I consider a Bayesian framework (with fixed known hyperparameters) with a weight prior that factorizes over hidden units $p(\mathbf{w}, \mathbf{W}) = \prod_k [p(W_k)p(\mathbf{w}_k)]$ and Gaussian input-to-hidden weights $\mathbf{w}_k \sim \mathcal{N}(0, \mathbf{\Sigma})$.

**From Bayesian NNs to GPs.** The prior over outputs for the Bayesian neural network is $p(\mathbf{f}) = \int d\mathbf{w}d\mathbf{W} p(\mathbf{w}, \mathbf{W}) \prod_i \delta(f(\mathbf{x}_i) - f(\mathbf{x}_i, \mathbf{w}, \mathbf{W}))$. In the infinite hidden unit limit, $K \to \infty$, when $p(W_k)$ has zero mean and finite, say unit variance, it follows from the central limit theorem (CLT) that the prior distribution converges to a Gaussian process $\mathbf{f} \sim \mathcal{N}(0, \mathbf{C})$ with kernel [1, 2]

$$\begin{aligned} C(\mathbf{x}, \mathbf{x}') &= \int d\mathbf{w}\, p(\mathbf{w})\, \Phi(\mathbf{w} \cdot \mathbf{x})\, \Phi(\mathbf{w} \cdot \mathbf{x}') \\ &= \frac{2}{\pi} \arcsin\left(\frac{\mathbf{x}^T \mathbf{\Sigma} \mathbf{x}'}{\sqrt{(1 + \mathbf{x}^T \mathbf{\Sigma} \mathbf{x})(1 + \mathbf{x}'^T \mathbf{\Sigma} \mathbf{x}')}}\right) . \end{aligned} \tag{3}$$

The rest of the paper deals with theoretical statistical mechanics analysis and simulations for GPs and Bayesian NNs learning tasks defined by either a NN or a GP. For the simulations, I use analytical GP learning (scaling like $\mathcal{O}(N^3)$) [4] and a TAP mean field algorithm for Bayesian NN.

## 2 Statistical mechanics of learning

The aim of the average case statistical mechanics analysis is to derive learning curves, i.e. the expected generalization error as a function of the number of training examples. The generalization error of the Bayes regressor $\langle f(\mathbf{x}) \rangle$ eq. (1) is

$$\epsilon_g = \langle \langle (y - \langle f(\mathbf{x}) \rangle)^2 \rangle \rangle , \qquad (4)$$

where double brackets $\langle \langle \ldots \rangle \rangle = \int \prod_i [dx_i dy_i p(\mathbf{x}_i, y_i)] \ldots$ denote an average over both training examples and the test example $(\mathbf{x}, y)$. Rather than using eq. (4) directly, $\epsilon_g$ will–as usually done–be derived from the average of the free energy $-\langle \langle \ln Z \rangle \rangle$, where the partition function is given by

$$Z = \mathbf{E_f} \frac{1}{\sqrt{2\pi\sigma^2}^N} \exp \left( -\frac{1}{2\sigma^2} \sum_i (y_i - f(\mathbf{x}_i))^2 \right) . \qquad (5)$$

I will not give many details of the actual calculations here since it is beyond the scope of the paper, but only outline some of the basic assumptions.

### 2.1 Gaussian processes

The calculation for Gaussian processes is given in another NIPS contribution [5]. The basic assumption made is that $y - f(\mathbf{x})$ becomes Gaussian with zero mean[1] under an average over the training example $y - f(\mathbf{x}) \sim \mathcal{N}(0, \langle \langle (y - f(\mathbf{x}))^2 \rangle \rangle)$. This assumption can be justified by the CLT when $f(\mathbf{x})$ is a sum of many random parts contributing on the same scale. Corrections to the Gaussian assumption may also be calculated [5]. The free energy may be written in term of a set of order parameters which is found by saddlepoint integration.

Assuming that the teacher is noisy $y = f_*(\mathbf{x}) + \eta$, $\langle \langle \eta^2 \rangle \rangle = \sigma_*^2$, the generalization error is given by the following equation which depends upon an orderparameter $\nu$

$$\epsilon_g = \frac{\sigma_*^2 + \langle \langle f_*^2(\mathbf{x}) \rangle \rangle - \partial_\nu (\nu^2 \hat{\mathbf{E}}_\mathbf{f} \langle \langle f(\mathbf{x}) f_*(\mathbf{x}) \rangle \rangle^2)}{1 + \lambda^2 \partial_\nu \hat{\mathbf{E}}_\mathbf{f} \langle \langle f^2(\mathbf{x}) \rangle \rangle / N} \qquad (6)$$

$$\nu = \frac{N}{\sigma^2 + \hat{\mathbf{E}}_f \langle \langle f^2(\mathbf{x}) \rangle \rangle} , \qquad (7)$$

where the new normalized measure $\hat{\mathbf{E}}_\mathbf{f} \ldots \propto \mathbf{E_f} \exp \left( -\nu \langle \langle f^2(\mathbf{x}) \rangle \rangle / 2 \right) \ldots$ has been introduced.

**Kernels in feature space.** By performing a Karhunen-Loeve expansion, $f(\mathbf{x})$ can be written as a linear perceptron with weights $\omega_\rho$ in a possible infinite feature space

$$f(\mathbf{x}) = \sum_\rho \omega_\rho \sqrt{\lambda_\rho} \phi_\rho(\mathbf{x}) , \qquad (8)$$

where the features $\phi_\rho(\mathbf{x})$ are orthonormal eigenvectors of the covariance function with eigenvalues $\lambda_\rho$: $\int d\mathbf{x}\, p(\mathbf{x}) C(\mathbf{x}', \mathbf{x}) \phi_\rho(\mathbf{x}) = \lambda_\rho \phi_\rho(\mathbf{x}')$ and $\int d\mathbf{x}\, p(\mathbf{x}) \phi_{\rho'}(\mathbf{x}) \phi_\rho(\mathbf{x}) = \delta_{\rho\rho'}$. The teacher $f_*(\mathbf{x})$ may also be expanded in terms of the the features:

$$f_*(\mathbf{x}) = \sum_\rho a_\rho \sqrt{\lambda_\rho} \phi_\rho(\mathbf{x}) , \qquad a_\rho \sqrt{\lambda_\rho} = \int d\mathbf{x} p(\mathbf{x}) f_*(\mathbf{x}) \phi_\rho(\mathbf{x}) .$$

Using the orthonormality the averages may be found: $\langle \langle f^2(\mathbf{x}) \rangle \rangle = \sum_\rho \lambda_\rho \omega_\rho^2$, $\langle \langle f(\mathbf{x}) f_*(\mathbf{x}) \rangle \rangle = \sum_\rho \lambda_\rho \omega_\rho a_\rho$ and $\langle \langle f_*^2(\mathbf{x}) \rangle \rangle = \sum_\rho \lambda_\rho a_\rho^2$. For a Gaussian process prior,

the prior over the weight is a spherical Gaussian $\boldsymbol{\omega} \sim \mathcal{N}(0, \mathbf{I})$. Averaging over $\boldsymbol{\omega}$, the saddlepoint equations can be written in terms of the number of examples $N$, the noise levels $\sigma^2$ and $\sigma_*^2$, the eigenvectors of the covariance function $\lambda_\rho$ and the teacher projections $a_\rho$:

$$\epsilon_g = \frac{N}{\nu} \left( \sigma_*^2 + \sum_\rho \frac{\lambda_\rho a_\rho^2}{(1 + \nu\lambda_\rho)^2} \right) \left( \sigma^2 + \sum_\rho \frac{\lambda_\rho}{(1 + \nu\lambda_\rho)^2} \right)^{-1} \tag{9}$$

$$\nu = N \left( \sigma^2 + \sum_\rho \frac{\lambda_\rho}{1 + \nu\lambda_\rho} \right)^{-1} \tag{10}$$

These eqs. are valid for a fixed teacher. However, eq. (9) may also be averaged over the distribution of teachers. In the *Bayes optimal* scenario, the teacher is sampled from the same prior as the student and $\sigma^2 = \sigma_*^2$. Thus $a_\rho \sim \mathcal{N}(0, \mathbf{I})$ implying $\overline{a_\rho^2} = 1$, where the average over the teacher is denoted by an overline. In this case the equations reduce to the Bayes optimal result first derived by Sollich [6]: $\epsilon_g = \epsilon_g^{\text{Bayes}} = N/\nu$.

**Learning finite nets.** Next, I consider the case where the teacher is the two-layer network $f_*(\mathbf{x}) = f(\mathbf{w}, \mathbf{W})$ and the GP student uses the infinite net kernel eq. (3). The average over the teacher corresponds to an average over the weight prior and since $\overline{f_*(\mathbf{x}) f_*(\mathbf{x}')} = C(\mathbf{x}, \mathbf{x}')$, I get

$$\overline{a_\rho^2}\lambda_\rho = \int d\mathbf{x} d\mathbf{x}' p(\mathbf{x}) p(\mathbf{x}') C(\mathbf{x}, \mathbf{x}') \phi_\rho(\mathbf{x}) \phi_\rho(\mathbf{x}') = \lambda_\rho, \tag{11}$$

where the eigenvalue equation and the orthonormality have been used. The theory therefore predicts that a GP student (with the infinite network kernel) will have the same learning curve *irrespectively of the number of hidden units* of the NN teacher. This result is a direct consequence of the Gaussian assumption made for the average over examples. However, what is more surprising is that it is found to be a very good approximation in simulations down to $K = 1$, i.e. a simple perceptron with a sigmoid non-linearity.

**Inner product kernels.** I specialize to inner product kernels $C(\mathbf{x}, \mathbf{x}') = c(\mathbf{x} \cdot \mathbf{x}'/d)$ and consider large input dimensionality $d$ and input components which are iid with zero mean and unit variance. The eigenvectors are products of the input components $\phi_\rho(\mathbf{x}) = \prod_{m \in \rho} x_m$ and are indexed by subsets of input indices, e.g. $\rho = \{1, 2, 42\}$ [3]. The eigenvalues are $\lambda_\rho = \frac{c^{|\rho|}(0)}{d^{|\rho|}}$ with degeneracy $n_{|\rho|} = \binom{d}{|\rho|} \approx d^{|\rho|}/|\rho|!$, where $|\rho|$ is the cardinality (in the example above $|\rho| = 3$). Plugging these results into eqs. (9) and (10), it follows that *to learn features that are order $s$ in the inputs, $\mathcal{O}(d^s)$ examples are needed.* The same behavior has been predicted for learning in SVMs [3].

The infinite net eq. (3) reduces to an inner product covariance function for $\mathbf{\Sigma} = T\mathbf{I}/d$ ($T$ controls the degree on non-linearity of the rule) and large $d$, $\mathbf{x} \cdot \mathbf{x} \approx d$:

$$C(\mathbf{x}, \mathbf{x}') = c(\mathbf{x} \cdot \mathbf{x}'/d) = \frac{2}{\pi} \arcsin \left( \frac{T\mathbf{x} \cdot \mathbf{x}'}{d(1 + T)} \right). \tag{12}$$

Figure 1 shows learning curves for GPs for the infinite network kernel. The mismatch between theory and simulations is expected to be due to $\mathcal{O}(1/d)$-corrections to the eigenvalues $\lambda_\rho$. The figure clearly shows that learning of the different order features takes place on different scales. The stars on the $\epsilon_g$-axis show the theoretical prediction of asymptotic error for $N = \mathcal{O}(d), \mathcal{O}(d^3), \dots$ (the teacher is an odd function).

## 2.2 Bayesian neural networks

The limit of large but finite NNs allows for efficient computation since the prior over functions can be approximated by a Gaussian. The hidden-to-output weights are for sim-

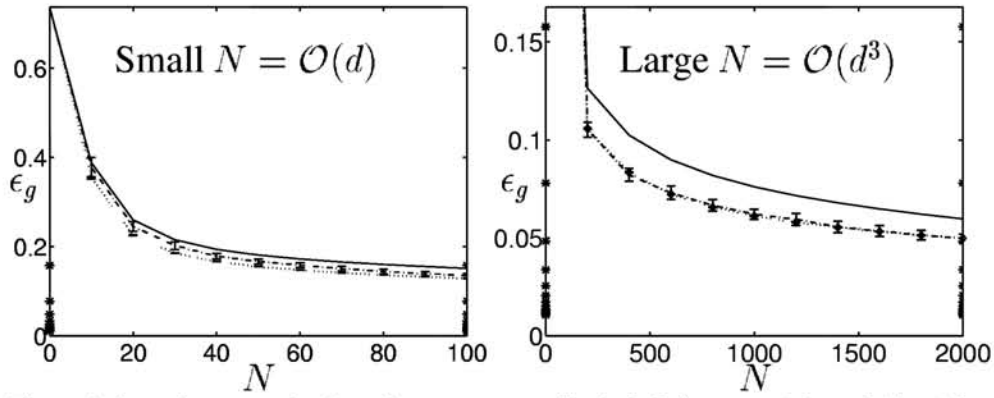

Figure 1: Learning curve for Gaussian processes with the infinite network kernel ($d = 10$, $T = 10$ and $\sigma^2 = 0.01$) for two scales of training examples. The full line is the the theoretical prediction for the Bayes optimal GP scenario. The two other curves (almost on top of each other as predicted by theory) are simulations for the Bayes optimal scenario (dotted line) and for GP learning a neural network with $K = 30$ hidden units (dash-dotted line).

plicity set to one and we introduce the 'fields' $h_k(\mathbf{x}) = \mathbf{w}_k \cdot \mathbf{x}$ and write the output as $f(\mathbf{x}, \mathbf{w}) = f(\mathbf{h}(\mathbf{x})) = \frac{1}{\sqrt{K}} \sum_k^K \Phi(h_k(\mathbf{x}))$, $\mathbf{h}(\mathbf{x}) = h_1(\mathbf{x}), \ldots, h_K(\mathbf{x})$. In the following, I discuss the TAP mean field algorithm used to find an approximation to the Bayes regressor and briefly the theoretical statistical mechanics analysis for the NN task.

**Mean field algorithm.** The derivation sketched here is a straightforward generalization of previous results for neural networks [7]. The basic *cavity* assumption [7, 8] is that for large $d$, $K$ and for a suitable input distribution, the predictive distribution $p(f(\mathbf{x})|D_N)$ is Gaussian:

$$p(f(\mathbf{x})|D_N) \approx \mathcal{N}(\langle f(\mathbf{x})\rangle, \langle f^2(\mathbf{x})\rangle - \langle f(\mathbf{x})\rangle^2) \ .$$

The predictive distribution for the fields $\mathbf{h}(\mathbf{x})$ is also assumed to be Gaussian

$$p(\mathbf{h}(\mathbf{x})|D_N) \approx \mathcal{N}(\langle \mathbf{h}(\mathbf{x})\rangle, \mathbf{V}) \ ,$$

where $\mathbf{V} = \langle \mathbf{h}(\mathbf{x})\mathbf{h}(\mathbf{x})^T\rangle - \langle \mathbf{h}(\mathbf{x})\rangle\langle \mathbf{h}(\mathbf{x})^T\rangle$. Using these assumptions, I get an approximate Bayes regressor

$$\langle f(\mathbf{x})\rangle \approx \frac{1}{\sqrt{K}} \sum_k \Phi\left(\frac{\langle h_k(\mathbf{x})\rangle}{\sqrt{1 + V_{kk}}}\right) \ . \tag{13}$$

To make predictions, we therefore need the two first moments of the weights since $\langle h_k(\mathbf{x})\rangle = \langle \mathbf{w}_k\rangle \cdot \mathbf{x}$ and $V_{kl} = \sum_{mn} x_m x_n (\langle w_{mk} w_{nl}\rangle - \langle w_{mk}\rangle\langle w_{nl}\rangle)$. We can simplify this in the large $d$ limit by taking the inputs to by iid with zero mean and unit variance: $V_{kl} \approx \langle \mathbf{w}_k \cdot \mathbf{w}_l\rangle - \langle \mathbf{w}_k\rangle \cdot \langle \mathbf{w}_l\rangle$. This approximation can be avoided at a substantial computational cost [8]. Furthermore, $\langle \mathbf{w}_k \cdot \mathbf{w}_l\rangle$ turns out equal to the prior covariance $\delta_{kl} T/d$ [7]. The following exact relation is obtained for the mean weights

$$\langle \mathbf{w}_k\rangle = \sum_i \alpha_{ki}\mathbf{x}_i \ , \quad \alpha_{ki} = \frac{\partial}{\partial\langle h_k(\mathbf{x}_i)\rangle} \ln p(y_i|D_N\backslash(\mathbf{x}_i, y_i)) \tag{14}$$

where

$$p(y_i|D_N\backslash(\mathbf{x}_i, y_i)) = \int d\mathbf{h}(\mathbf{x}_i)\, p(y_i|\mathbf{h}(\mathbf{x}_i))\, p(\mathbf{h}(\mathbf{x}_i)|D_N\backslash(\mathbf{x}_i, y_i)) \ .$$

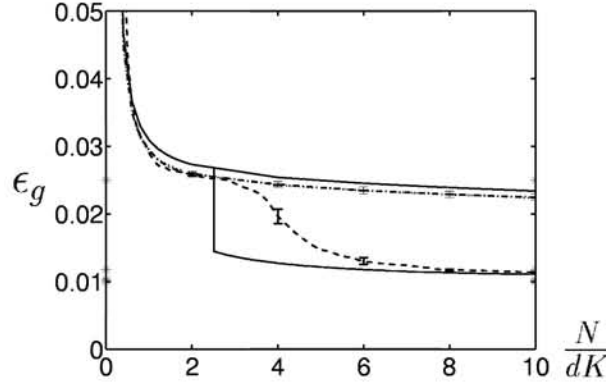

Figure 2: . Learning curves for Bayesian NNs and GPs. The dashed line is simulations for the TAP mean field algorithm ($d = 30$, $K = 5$, $T = 1$ and $\sigma^2 = 0.01$) learning a corresponding NN task, i.e. an approximation to the Bayes optimal scenario. The dash-dotted line is the simulations for GPs learning the NN task. Virtually on top of that curve is the curve for Bayes optimal GP scenario (dotted line). The full lines are the theoretical prediction. Up to $N = N_c = 2.51dK$, the learning curves for Bayesian NNs and GPs coincide. At $N_c$, the statistical mechanics theory predicts a first order transition to a specialized solution for the NN Bayes optimal scenario (lower full line).

$p(y_i | \mathbf{h}(\mathbf{x}_i))$ is the Likelihood and $p(\mathbf{h}(\mathbf{x}_i) | D_N \backslash (\mathbf{x}_i, y_i))$ is a predictive distribution for $\mathbf{h}(\mathbf{x}_i)$ for a training set where the $i$th example has been left out. In accordance with above, I assume $p(\mathbf{h}(\mathbf{x}_i) | D_N \backslash (\mathbf{x}_i, y_i)) \approx \mathcal{N}(\langle \mathbf{h}(\mathbf{x}_i) \rangle_{\backslash i}, \mathbf{V})$. Finally, generalizing the relation found in Refs. [7, 8], I can relate the reduced mean to the full posterior mean:

$$\langle h_k(\mathbf{x}_i) \rangle_{\backslash i} = \langle h_k(\mathbf{x}_i) \rangle - \sum_l V_{kl} \alpha_{li}$$

to express everything in terms of $\langle \mathbf{w}_k \rangle$ and $\alpha_{ki}$, $k = 1, \ldots, K$ and $i = 1, \ldots, N$.

The mean field eqs. are solved by iteration in $\alpha_{ki}$ and $\langle w_{mk} \rangle$ following the recipe given in Ref. [8]. The algorithm is tested using a teacher sampled from the NN prior, i.e. the Bayes optimal scenario. Two types of solutions are found: a linear symmetric and a non-linear specialized. In the symmetric solution, $\langle \mathbf{w}_k \rangle = \langle \mathbf{w}_l \rangle$ and $\langle \mathbf{w}_k \rangle \cdot \langle \mathbf{w}_k \rangle = \mathcal{O}(T/dK)$. This means that the machine is linear (when $T \ll K$). For $N = \mathcal{O}(dK)$, a transition to a specialized solution occurs, where each $\langle \mathbf{w}_k \rangle$, $k = 1, \ldots, K$, aligns to a distinct weight vector of the teacher and $\langle \mathbf{w}_k \rangle \cdot \langle \mathbf{w}_k \rangle = \mathcal{O}(T/d)$. The Bayesian student thus learns the linear features for $N = \mathcal{O}(d)$. However, unlike the GP, it learns all of the remaining non-linear features for $N = \mathcal{O}(dK)$. The resulting empirical learning curve averaged over 25 independent runs is shown in figure 2. It turned out that setting $\langle h_k(\mathbf{x}_i) \rangle_{\backslash i} = \langle h_k(\mathbf{x}_i) \rangle$ was a necessary heuristic in order to find the specialized solution. The transition to the specialized solution–although very abrupt for the individual run–is smeared out because it occurs at different $N$ for each run.

**The theoretical learning curve** is also shown in figure 2. It has been derived by generalizing the results of Ref. [9] for the Gibbs algorithm to the Bayes optimal scenario. The picture that emerges is in accordance with the empirical findings. The transition to the specialized solution is predicted to be first order, i.e. with a discontinuous jump in the relevant order parameters at the number of examples $N_c(\sigma^2, T)$, where the specialized solution becomes the physical solution (i.e. the lowest free energy solution).

The mean field algorithm cannot completely reproduce the theoretical predictions because the solution gets trapped in the meta-stable symmetric solution. This is often observed

for first order transitions and should also be observable in the Monte Carlo approach to Bayesian NNs [1].

## 3   Discussion

Learning a finite two-layer regression NN using (1) the Bayes optimal algorithm and (2) the Bayes optimal algorithm for an infinite network (implemented by a GP) is compared. It is found that the Bayes optimal algorithm can have a very superior performance.

This can be explained as an entropic effect: The infinite network will–although the correct finite network solution is included a priori– have a vanishing probability of finding this solution. The finite network on the other hand is much more constraint wrt the functions it implements. It can thus–even in the Bayesian setting–give a great pay off to limit complexity.

For $d$-dimensional inner product kernel with iid input distribution, it is found that it in general requires $\mathcal{O}(d^s)$ training examples to learn features of $\mathcal{O}(s)$. Unpublished results and [3] show that these conclusions remain true also for SVM and GP classification.

For SVM hand-written digit recognition, fourth order kernels give good results in practise. Since $N = \mathcal{O}(10^4) - \mathcal{O}(10^5)$, it can be concluded that the 'effective' dimension, $d_{\text{effective}} = \mathcal{O}(10)$ against typically $d = 400$, i.e. some inputs must be very correlated and/or carry very little information. It could therefore be interesting to develop methods to measure the effective dimension and to extract the important lower dimensional features rather than performing the classification directly from the images.

### Acknowledgments

I am thankful to Manfred Opper for valuable discussions and for sharing his results with me and to Klaus-Robert Müller for discussions at NIPS. This research is supported by the Swedish Foundation for Strategic Research.

## Footnotes

*`http://www.thep.lu.se/tf2/staff/winther/`

[1]Generalization to non-zero mean is straightforward.

## References

[1] R. Neal, *Bayesian Learning for Neural Networks*, Lecture Notes in Statistics, Springer (1996).

[2] C. K. I. Williams, Computing with Infinite Networks, in *Neural Information Processing Systems 9*, Eds. M. C. Mozer, M. I. Jordan and T. Petsche, 295-301, MIT Press (1997).

[3] R. Dietrich, M. Opper and H. Sompolinsky, Statistical Mechanics of Support Vector Machines, Phys. Rev. Lett. **82**, 2975-2978 (1999).

[4] C. K. I. Williams and C. E. Rasmussen, Gaussian Processes for Regression , In Advances in Neural Information Processing Systems 8 (NIPS'95). Eds. D. S. Touretzky, M. C. Mozer and M. E. Hasselmo, 514-520, MIT Press (1996).

[5] D. Malzahn and M. Opper, In this volume.

[6] P. Sollich, Learning Curves for Gaussian Processes, In Advances in Neural Information Processing Systems 11 (NIPS'98), Eds. M. S. Kearns, S. A. Solla, and D. A. Cohn, 344-350, MIT Press (1999).

[7] M. Opper and O. Winther, Mean Field Approach to Bayes Learning in Feed-Forward Neural Networks, Phys. Rev. Lett. **76**, 1964-1967 (1996).

[8] M. Opper and O. Winther, Gaussian Processes for Classification: Mean Field Algorithms, Neural Computation **12**, 2655-2684 (2000).

[9] M. Ahr, M. Biehl and R. Urbanczik, Statistical physics and practical training of soft-committee machines Eur. Phys. J. **B 10**, 583 (1999).
